# Inferring Network Structure from Co-Occurrences

**Michael G. Rabbat**
Electrical and Computer Eng.
University of Wisconsin
Madison, WI 53706
rabbat@cae.wisc.edu

**Mário A.T. Figueiredo**
*Instituto de Telecomunicações*
*Instituto Superior Técnico*
Lisboa, Portugal
mtf@lx.it.pt

**Robert D. Nowak**
Electrical and Computer Eng.
University of Wisconsin
Madison, WI 53706
nowak@ece.wisc.edu

## Abstract

We consider the problem of inferring the structure of a network from co-occurrence data: observations that indicate which nodes occur in a signaling pathway but do not directly reveal node order within the pathway. This problem is motivated by network inference problems arising in computational biology and communication systems, in which it is difficult or impossible to obtain precise time ordering information. Without order information, every permutation of the activated nodes leads to a different feasible solution, resulting in combinatorial explosion of the feasible set. However, physical principles underlying most networked systems suggest that not all feasible solutions are equally likely. Intuitively, nodes that co-occur more frequently are probably more closely connected. Building on this intuition, we model path co-occurrences as randomly shuffled samples of a random walk on the network. We derive a computationally efficient network inference algorithm and, via novel concentration inequalities for importance sampling estimators, prove that a polynomial complexity Monte Carlo version of the algorithm converges with high probability.

## 1 Introduction

The study of complex networked systems is an emerging field impacting nearly every area of engineering and science, including the important domains of biology, cognitive science, sociology, and telecommunications. Inferring the structure of signalling networks from experimental data precedes any such analysis and is thus a basic and fundamental task. Measurements which directly reveal network structure are often beyond experimental capabilities or are excessively expensive. This paper addresses the problem of inferring the structure of a network from co-occurrence data: observations which indicate nodes that are activated in each of a set of signaling pathways but do not directly reveal the order of nodes within each pathway. Co-occurrence observations arise naturally in a number of interesting contexts, including biological and communication networks, and networks of neuronal colonies.

Biological signal transduction networks describe fundamental cell functions and responses to environmental stress [1]. Although it is possible to test for individual, localized interactions between gene pairs, this approach (called genetic epistatic analysis) is expensive and time-consuming. High-throughput measurement techniques such as microarrays have successfully been used to identify the components of different signal transduction pathways [2]. However, microarray data only reflects order information at a very coarse, unreliable level. Developing computational techniques for inferring pathway orders is a largely unexplored research area [3].

A similar problem has been studied in telecommunication networks [4]. In this context, each path corresponds to a transmission between an origin and destination. The origin and destination are observed, in addition to the activated switches/routers carrying the transmission through the network.

However, due to the geographically distributed nature of the measurement infrastructure and the rapidity at which transmissions are completed, it is not possible to obtain precise ordering information.

Another exciting potential application arises in neuroimaging [5, 6]. Functional magnetic resonance imaging provides images of brain activity with high spatial resolution but has relatively poor temporal resolution. Treating distinct brain regions as nodes in a functional brain network that co-activate when a subject performs different tasks may lead to a similar network inference problem.

Given a collection of co-occurrences, a feasible network (consistent with the observations) is easily obtained by assigning an order to the elements of each co-occurrence, thereby specifying a path through the hypothesized network. Since any arbitrary order of each co-occurrence leads to a feasible network, the number of feasible solutions is proportional to the number of permutations of all the co-occurrence observations. Consequently we are faced with combinatorial explosion of the feasible set, and without additional assumptions or side information there is no reason to prefer one particular feasible network over the others. See the supplementary document [7] for further discussion.

Despite the apparent intractability of the problem, physical principles governing most networks suggest that not all feasible solutions are equally plausible. Intuitively, nodes that co-occur more frequently are more likely to be connected in the underlying network. This intuition has been used as a stepping stone by recent approaches proposed in the context of telecommunications [4], and in learning networks of collaborators [8]. However, because of their heuristic nature, these approaches do not produce easily interpreted results and do not readily lend themselves to analysis or to the incorporation of side information.

In this paper, we model co-occurrences as randomly permuted samples of a random walk on the underlying network. The random permutation accounts for lack of observed order. We refer to this process as the shuffled Markov model. In this framework, network inference amounts to maximum likelihood estimation of the parameters governing the random walk (initial state distribution and transition matrix). Direct maximization is intractable due to the highly non-convex log-likelihood function and exponential feasible set arising from simultaneously considering all permutations of all co-occurrences. Instead, we derive a computationally efficient EM algorithm, treating the random permutations as hidden variables. In this framework the likelihood factorizes with respect to each pathway/observation, so that the computational complexity of the EM algorithm is determined by the E-step which is only exponential in the longest path. In order to handle networks with long paths, we propose a Monte Carlo E-step based on a simple, linear complexity importance sampling scheme. Whereas the exact E-step has computational complexity which is exponential in path length, we prove that a polynomial number of importance samples suffices to retain desirable convergence properties of the EM algorithm with high probability. In this sense, our Monte Carlo EM algorithm breaks the curse of dimensionality using randomness.

It is worth noting that the approach described here differs considerably from that of learning the structure of a directed graphical model or Bayesian network [9, 10]. The aim of graphical modelling is to find a graph corresponding to a factorization of a high-dimensional distribution which predicts the observations well. These probabilistic models do not directly reflect physical structures, and applying such an approach to co-occurrences would ignore physical constraints inherent to the observations: co-occurring vertices must lie along a path in the network.

## 2 Model Formulation and EM Algorithm

### 2.1 The Shuffled Markov Model

We model a network as a directed graph $G = (V, E)$, where $V = \{1, \ldots, |V|\}$ is the vertex (node) set and $E \subseteq V^2$ is the set of edges (direct connections between vertices). An observation, $\mathbf{y} \subset V$, is a subset of vertices co-activated when a particular stimulus is applied to the network (*e.g.*, collection of signaling proteins activated in response to an environmental stress). Given a set of $T$ observations, $\mathcal{Y} = \{\mathbf{y}^{(1)}, \ldots, \mathbf{y}^{(T)}\}$, each corresponding to a path, where $\mathbf{y}^{(m)} = \{y_1^{(m)}, \ldots, y_{N_m}^{(m)}\}$, we say that a graph $(V, E)$ is feasible w.r.t. $\mathcal{Y}$ if for each $\mathbf{y}^{(m)} \in \mathcal{Y}$ there is an ordered path $\mathbf{z}^{(m)} = (z_1^{(m)}, \ldots, z_{N_m}^{(m)})$ and a permutation $\boldsymbol{\tau}^{(m)} = (\tau_1^{(m)}, \ldots, \tau_{N_m}^{(m)})$ such that $z_t^{(m)} = y_{\tau_t^{(m)}}^{(m)}$, and $(z_{t-1}, z_t) \in E$, for $t = 2, \ldots, N_m$.

The (unobserved) ordered paths, $\mathcal{Z} = \{\mathbf{z}^{(1)}, ..., \mathbf{z}^{(T)}\}$, are modelled as $T$ independent samples of a first-order Markov chain with state set $V$. The Markov chain is parameterized by the initial state distribution $\boldsymbol{\pi}$ and the (stochastic) transition matrix $\mathbf{A}$. We assume that the support of the transition matrix is determined by the adjacency structure of the graph; *i.e.*, $A_{i,j} > 0 \Leftrightarrow (i, j) \in E$. Each observation $\mathbf{y}^{(m)}$ results from shuffling the elements of $\mathbf{z}^{(m)}$ via an unobserved permutation $\boldsymbol{\tau}^{(m)}$, drawn uniformly from $\mathbb{S}_{N_m}$ (the set of all permutations of $N_m$ objects); *i.e.*, $z_t^{(m)} = y_{\tau_t^{(m)}}^{(m)}$, for $t = 1, \ldots, N_m$. All the $\boldsymbol{\tau}^{(m)}$ are assumed mutually independent and independent of all the $\mathbf{z}^{(m)}$. Under this model, the log-likelihood of the set of observations $\mathcal{Y}$ is

$$\log P[\mathcal{Y}|\mathbf{A}, \boldsymbol{\pi}] = \sum_{m=1}^{T} \left( \log \left( \sum_{\boldsymbol{\tau} \in \mathbb{S}_{N_m}} P[\mathbf{y}^{(m)}|\boldsymbol{\tau}, \mathbf{A}, \boldsymbol{\pi}] \right) - \log(N_m!) \right). \quad (1)$$

where $P[\mathbf{y}|\boldsymbol{\tau}, \mathbf{A}, \boldsymbol{\pi}] = \pi_{y_{\tau_1}} \prod_{t=2}^{N} A_{y_{\tau_{t-1}}, y_{\tau_t}}$, and network inference consists in computing the maximum likelihood (ML) estimates $(\mathbf{A}_{\mathrm{ML}}, \boldsymbol{\pi}_{\mathrm{ML}}) = \arg\max_{\mathbf{A}, \boldsymbol{\pi}} \log P[\mathcal{Y}|\mathbf{A}, \boldsymbol{\pi}]$. With the ML estimates in hand, we may determine the most likely permutation for each $\mathbf{y}^{(m)}$ and obtain a feasible reconstruction from the ordered paths. In general, $\log P[\mathcal{Y}|\mathbf{A}, \boldsymbol{\pi}]$ is a non-concave function of $(\mathbf{A}, \boldsymbol{\pi})$, so finding $(\mathbf{A}_{\mathrm{ML}}, \boldsymbol{\pi}_{\mathrm{ML}})$ is not easy. Next, we derive an EM algorithm for this purpose, by treating the permutations as missing data.

## 2.2 EM Algorithm

Let $\mathbf{w}^{(m)} = (\mathbf{w}_1^{(m)}, ..., \mathbf{w}_{N_m}^{(m)})$ be a binary representation of $\mathbf{z}^{(m)}$, defined by $\mathbf{w}_t^{(m)} = (w_{t,1}^{(m)}, ..., w_{t,|V|}^{(m)}) \in \{0, 1\}^{|V|}$, with $(w_{t,i}^{(m)} = 1) \Leftrightarrow (z_t^{(m)} = i)$; let $\mathcal{W} = \{\mathbf{w}^{(1)}, ..., \mathbf{w}^{(T)}\}$. Let $\mathcal{X} = \{\mathbf{x}^{(1)}, \ldots, \mathbf{x}^{(T)}\}$ be the binary representation for $\mathcal{Y}$, defined in a similar way: $\mathbf{x}^{(m)} = (\mathbf{x}_1^{(m)}, ..., \mathbf{x}_{N_m}^{(m)})$, where $\mathbf{x}_t^{(m)} = (x_{t,1}^{(m)}, ..., x_{t,|V|}^{(m)}) \in \{0, 1\}^{|V|}$, with $(x_{t,i}^{(m)} = 1) \Leftrightarrow (y_t^{(m)} = i)$. Finally, let $\mathcal{R} = \{\mathbf{r}^{(1)}, \ldots, \mathbf{r}^{(T)}\}$ be the collection of permutation matrices corresponding to $\mathcal{T} = \{\boldsymbol{\tau}^{(1)}, \ldots, \boldsymbol{\tau}^{(T)}\}$; *i.e.*, $(r_{t,t'}^{(m)} = 1) \Leftrightarrow (\tau_t^{(m)} = t')$. With this notation in place, the complete log-likelihood can be written as $\log P[\mathcal{X}, \mathcal{R}|\mathbf{A}, \boldsymbol{\pi}] = \log P[\mathcal{X}|\mathcal{R}, \mathbf{A}, \boldsymbol{\pi}] + \log P[\mathcal{R}]$, where

$$\log P[\mathcal{X}|\mathcal{R}, \mathbf{A}, \boldsymbol{\pi}] = \sum_{m=1}^{T} \log P[\mathbf{x}^{(m)}|\mathbf{r}^{(m)}, \mathbf{A}, \boldsymbol{\pi}]$$

$$= \sum_{m=1}^{T} \sum_{i,j=1}^{|V|} \sum_{t',t''=1}^{N_m} \sum_{t=2}^{N_m} r_{t,t'}^{(m)} r_{t-1,t''}^{(m)} x_{t'',i}^{(m)} x_{t',j}^{(m)} \log A_{i,j} + \sum_{m=1}^{T} \sum_{i=1}^{|V|} \sum_{t'=1}^{N_m} r_{1,t'}^{(m)} x_{t',i}^{(m)} \log \pi_i, \quad (2)$$

and $P[\mathcal{R}]$ is the probability of the set of permutations, which is constant and thus dropped, since the permutations are independent and equiprobable.

The EM algorithm proceeds by (the E-step) computing $Q\left(\mathbf{A}, \boldsymbol{\pi}; \mathbf{A}^k, \boldsymbol{\pi}^k\right) = E\left[\log P[\mathcal{X}, \mathcal{R}|\mathbf{A}, \boldsymbol{\pi}] \,|\, \mathcal{X}, \mathbf{A}^k, \boldsymbol{\pi}^k\right]$, the expected value of $\log P[\mathcal{X}, \mathcal{R}|\mathbf{A}, \boldsymbol{\pi}]$ w.r.t. the missing $\mathcal{R}$, conditioned on the observations and on the current model estimate $(\mathbf{A}^k, \boldsymbol{\pi}^k)$. Examining $\log P[\mathcal{X}, \mathcal{R}|\mathbf{A}, \boldsymbol{\pi}]$ reveals that it is linear w.r.t. simple functions of $\mathcal{R}$: **(a)** the first row of each $\mathbf{r}^{(m)}$, *i.e.*, $r_{1,t'}^{(m)}$; **(b)** sums of transition indicators, *i.e.*, $\alpha_{t',t''}^{(m)} \equiv \sum_{t=2}^{N_m} r_{t,t'}^{(m)} r_{t-1,t''}^{(m)}$. Consequently, the E-step reduces to computing the conditional expectations of $r_{1,t'}^{(m)}$ and $\alpha_{t',t''}^{(m)}$, denoted $\bar{r}_{1,t'}^{(m)}$ and $\bar{\alpha}_{t',t''}^{(m)}$, respectively, and plugging them into the complete log-likelihood (2), which yields $Q\left(\mathbf{A}, \boldsymbol{\pi}; \mathbf{A}^k, \boldsymbol{\pi}^k\right)$.

Since the permutations are (a priori) equiprobable, we have $P[\mathbf{r}^{(m)}] = (N_m!)^{-1}$, $P[r_{1,t'}^{(m)} = 1] = (N_m - 1)!/N_m! = 1/N_m$, and $P[\mathbf{r}^{(m)}|r_{1,t'}^{(m)} = 1] = 1/(N_m - 1)!$. Using these facts, the mutual independence among different observations, and Bayes law, it is not hard to show that

$$\bar{r}_{1,t'}^{(m)} = \frac{\gamma_{t'}^{(m)}}{\sum_{t'=1}^{N_m} \gamma_{t'}^{(m)}} \qquad \text{with} \qquad \gamma_{t'}^{(m)} = \sum_{\mathbf{r}: r_{1,t'}=1} P[\mathbf{x}^{(m)}|\mathbf{r}, \mathbf{A}^k, \boldsymbol{\pi}^k], \quad (3)$$

where each term $P\big[\mathbf{x}^{(m)}\big|\mathbf{r},\mathbf{A}^k,\boldsymbol{\pi}^k\big]$ is easily computed after using $\mathbf{r}$ to "unshuffle" $\mathbf{x}^{(m)}$:

$$P\big[\mathbf{x}^{(m)}\big|\mathbf{r},\mathbf{A}^k,\boldsymbol{\pi}^k\big] = P\big[\mathbf{y}^{(m)}\big|\boldsymbol{\tau},\mathbf{A}^k,\boldsymbol{\pi}^k\big] = \pi^k_{y^{(m)}_{\tau_1}}\prod_{t=2}^{N_m}A^k_{y^{(m)}_{\tau_{t-1}},y^{(m)}_{\tau_t}}.$$

The computation of $\bar{\alpha}^{(m)}_{t',t''}$ is similar to that of $\bar{r}^{(m)}_{1,t'}$; the key observations are that $P[r^{(m)}_{t,t'}r^{(m)}_{t-1,t''}=1] = (N_m-2)!/N_m!$ and $P[\mathbf{r}^{(m)}|r^{(m)}_{t,t'}r^{(m)}_{t-1,t''}=1] = 1/(N_m-2)!$, leading to

$$\bar{\alpha}^{(m)}_{t',t''} = \frac{\gamma^{(m)}_{t',t''}}{\sum_{t'=1}^{N_m}\gamma^{(m)}_{t'}}, \qquad \text{with} \qquad \gamma^{(m)}_{t',t''} = \sum_{\mathbf{r}}P[\mathbf{x}^{(m)}|\mathbf{r},\mathbf{A}^k,\boldsymbol{\pi}^k]\sum_{t=2}^{N_m}r_{t,t'}r_{t-1,t''}. \tag{4}$$

Computing $\{\bar{r}^{(m)}_{1,t'}\}$ and $\{\bar{\alpha}^{(m)}_{t',t''}\}$ requires $O\big(N_m!\big)$ operations. For large $N_m$, this is a heavy load; in Section 3, we describe a sampling approach for computing approximations to $\bar{r}_{1,t'}$ and $\bar{\alpha}_{t',t''}$.

Maximization of $Q\big(\mathbf{A},\boldsymbol{\pi};\mathbf{A}^k,\boldsymbol{\pi}^k\big)$ w.r.t. $\mathbf{A}$ and $\boldsymbol{\pi}$, under the normalization constraints, leads to the M-step:

$$A^{k+1}_{i,j} = \frac{\sum_{m=1}^{T}\sum_{t',t''=1}^{N_m}\bar{\alpha}^{(m)}_{t',t''}x^{(m)}_{t',i}x^{(m)}_{t',j}}{\sum_{j=1}^{|S|}\sum_{m=1}^{T}\sum_{t',t''=1}^{N_m}\bar{\alpha}^{(m)}_{t',t''}x^{(m)}_{t'',i}x^{(m)}_{t',j}} \quad \text{and} \quad \pi^{k+1}_i = \frac{\sum_{m=1}^{T}\sum_{t'=1}^{N_m}\bar{r}^{(m)}_{1,t'}x^{(m)}_{t',i}}{\sum_{i=1}^{|S|}\sum_{m=1}^{T}\sum_{t'=1}^{N_m}\bar{r}^{(m)}_{1,t'}x^{(m)}_{t',i}}. \tag{5}$$

Standard convergence results for the EM algorithm due to Boyles and Wu [11, 12] guarantee that the sequence $\{(\mathbf{A}^k,\boldsymbol{\pi}^k)\}$ converges monotonically to a local maximum of the likelihood.

### 2.3 Handling Known Endpoints

In some applications, (one or both of) the endpoints of each path are known and only the internal nodes are shuffled. For example, in telecommunications problems, the origin and destination of each transmission are known, but not the network connectivity. In estimating biological signal transduction pathways, a physical stimulus (*e.g.*, hypotonic shock) causes a sequence of protein interactions, resulting in another observable physical response (*e.g.*, a change in cell wall structure); in this case, the stimulus and response act as fixed endpoints, the goal is to infer the order of the sequence of protein interactions. Knowledge of the endpoints of each path imposes the constraints $r^{(m)}_{1,1}=1$ and $r^{(m)}_{N_m,N_m}=1$. Under the first constraint, estimates of the initial state probabilities are simply given by $\pi_i = \frac{1}{T}\sum_{m=1}^{T}x^{(m)}_{1,i}$. Thus, EM only needs to be used to estimate $\mathbf{A}$. In this setup, the E-step has a similar form as (4) but with sums over $\mathbf{r}$ replaced by sums over permutation matrices satisfying $r_{1,1}=1$ and $r_{N,N}=1$. The M-step update for $\mathbf{A}^{k+1}$ remains unchanged.

## 3 Large Scale Inference via Importance Sampling

For long paths, the combinatorial nature of the exact E-step – summing over all permutations of each sequence in (3) and (4) – may render exact computation intractable. This section presents a Monte Carlo importance sampling (see, *e.g.*, [13]) version of the E-step, along with finite sample bounds guaranteeing that a polynomial complexity Monte Carlo EM algorithm retains desirable convergence properties of the EM algorithm; *i.e.*, monotonic convergence to a local maximum.

### 3.1 Monte Carlo E-Step by Importance Sampling

To lighten notation in this section we drop the superscripts from $(\mathbf{A}^k,\boldsymbol{\pi}^k)$, using simply $(\mathbf{A},\boldsymbol{\pi})$ for the current parameter estimates. Moreover, since the statistics $\bar{\alpha}^{(m)}_{t',t''}$ and $\bar{r}^{(m)}_{1,t'}$ depend only on the $m$th co-activation observation, $\mathbf{y}^{(m)}$, we focus on a particular length-$N$ path observation $\mathbf{y} = (y_1, y_2, \ldots, y_N)$ and drop the superscript $(m)$.

A naïve Monte Carlo approximation would be based on random permutations sampled from the uniform distribution on $\mathbb{S}_N$. However, the reason we resort to approximation techniques in the first

place is that $\mathbb{S}_N$ is large, but typically only a small fraction of its elements have non-negligible posterior probability, $P[\boldsymbol{\tau}|\mathbf{y}, \mathbf{A}, \boldsymbol{\pi}]$. Although we would ideally sample directly from the posterior, this would require determining its value for all $N!$ permutations. Instead, we propose the following sequential scheme for sampling a permutation using the current parameter estimates, $(\mathbf{A}, \boldsymbol{\pi})$. To ensure the same element is not sampled twice we introduce a vector of binary flags, $\mathbf{f} = (f_1, f_2, \ldots, f_{|V|}) \in \{0, 1\}^{|V|}$. Given a probability distribution $\mathbf{p} = (p_1, p_2, \ldots, p_{|V|})$ on the vertex set, $V$, denote by $\mathbf{p}|\mathbf{f}$ the restriction of $\mathbf{p}$ to those elements $i \in V$ for which $f_i = 1$; *i.e.*,

$$(\mathbf{p}|\mathbf{f})_i = \frac{p_i f_i}{\sum_{j=1}^{|V|} p_j f_j}, \qquad \text{for } i = 1, 2, \ldots, |V|. \tag{6}$$

Our sampling scheme proceeds as follows:

**Step 1:** Initialize $\mathbf{f}$ so that $f_i = 1$ if $y_t = i$ for some $t = 1, \ldots, N$, and $f_i = 0$ otherwise.
Sample an element $v$ from $V$ according to the distribution $\boldsymbol{\pi}|\mathbf{f}$ on $V$.
Find $t$ such that $y_t = v$. Set $\tau_1 = t$.
Set $f_v = 0$ to prevent $y_t$ from being sampled again (ensure $\boldsymbol{\tau}$ is a permutation). Set $i = 2$.

**Step 2:** Let $\mathbf{A}_v$ denote the $v$th row of the transition matrix.
Sample an element $v'$ from $V$ according to the distribution $\mathbf{A}_v|\mathbf{f}$ on $V$.
Find $t$ such that $y_t = v'$. Set $\tau_i = t$. Set $f_{v'} = 0$.

**Step 3:** While $i < N$, update $v \leftarrow v'$ and $i \leftarrow i + 1$ and repeat Step 2; otherwise, stop.

Repeating this sampling procedure $L$ times yields a collection of iid permutations $\boldsymbol{\tau}^1, \boldsymbol{\tau}^2, \ldots, \boldsymbol{\tau}^L$, where the superscript now identifies the sample number; the corresponding permutation matrices are $\mathbf{r}^1, \mathbf{r}^2, \ldots, \mathbf{r}^L$. Samples generated according to the scheme described above are drawn from a distribution $R[\boldsymbol{\tau}|\mathbf{x}, \mathbf{A}, \boldsymbol{\pi}]$ on $\mathbb{S}_N$ which is different from the posterior $P[\boldsymbol{\tau}|\mathbf{x}, \mathbf{A}, \boldsymbol{\pi}]$. Importance sample estimates correct for this disparity and are given by the expressions

$$\widehat{r}_{1,t'} = \frac{\sum_{\ell=1}^L u_\ell r_{1,t'}^\ell}{\sum_{\ell=1}^L u_\ell} \quad \text{and} \quad \widehat{\alpha}_{t',t''} = \frac{\sum_{\ell=1}^L u_\ell \sum_{t=2}^N r_{t,t'}^\ell r_{t-1,t''}^\ell}{\sum_{\ell=1}^L u_\ell}, \tag{7}$$

where the correction factor (or weight) for sample $\mathbf{r}^\ell$ is given by

$$u_\ell = \frac{P[\mathbf{r}^\ell|\mathbf{x}, \mathbf{A}, \boldsymbol{\pi}]}{R[\mathbf{r}^\ell|\mathbf{x}, \mathbf{A}, \boldsymbol{\pi}]} = \frac{P[\boldsymbol{\tau}^\ell|\mathbf{y}, \mathbf{A}, \boldsymbol{\pi}]}{R[\boldsymbol{\tau}^\ell|\mathbf{y}, \mathbf{A}, \boldsymbol{\pi}]} = \prod_{t=2}^N \sum_{t'=t}^N A_{y_{\tau_{t-1}^\ell}, y_{\tau_{t'}^\ell}}. \tag{8}$$

A detailed derivation of the exact form of the induced distribution, $R$, and the correction factor, $u_\ell$, based on the sequential nature of the sampling scheme, along with further discussion and comparison with alternative sampling schemes can be found in the supplementary document [7]. In fact, terms in the product (8) are readily available as a byproduct of Step 2 (denominator of $\mathbf{A}_v|\mathbf{f}$).

## 3.2 Monotonicity and Convergence

Standard EM convergence results directly apply when the exact E-step is used [11, 12]. Let $\boldsymbol{\theta}^k = (\mathbf{A}^k, \boldsymbol{\pi}^k)$. By choosing $\boldsymbol{\theta}^{k+1}$ according to (5) we have $\boldsymbol{\theta}^{k+1} = \arg\max_{\boldsymbol{\theta}} Q(\boldsymbol{\theta}; \boldsymbol{\theta}^k)$, and the *monotonicity property*, $Q(\boldsymbol{\theta}^{k+1}; \boldsymbol{\theta}^k) \geq Q(\boldsymbol{\theta}^k; \boldsymbol{\theta}^k)$, is satisfied. Together with the fact that the marginal log-likelihood (1) is continuous in $\boldsymbol{\theta}$ and bounded above, the monotonicity property guarantees that the exact EM iterates converge monotonically to a local maximum of $\log P[\mathcal{Y}|\boldsymbol{\theta}]$.

When the Monte Carlo E-step is used, we no longer have monotonicity since now the M-step solves $\widehat{\boldsymbol{\theta}}^{k+1} = \arg\max_{\boldsymbol{\theta}} \widehat{Q}(\boldsymbol{\theta}; \widehat{\boldsymbol{\theta}}^k)$, where $\widehat{Q}$ is defined analogously to $Q$ but with $\bar{\alpha}_{t',t''}^{(m)}$ and $\bar{r}_{1,t'}^{(m)}$ replaced by $\widehat{\alpha}_{t',t''}^{(m)}$ and $\widehat{r}_{1,t'}^{(m)}$; for monotonicity we need $Q(\widehat{\boldsymbol{\theta}}^{k+1}; \widehat{\boldsymbol{\theta}}^k) \geq Q(\widehat{\boldsymbol{\theta}}^k; \widehat{\boldsymbol{\theta}}^k)$. To assure the *Monte Carlo EM algorithm* (MCEM) converges, the number of importance samples, $L$, must be chosen carefully so that $\widehat{Q}$ approximates $Q$ well enough; otherwise the MCEM may be swamped with error.

Recently, Caffo et al. [14] have proposed a method, based on central limit theorem-like arguments, for automatically adapting the number of Monte Carlo samples used at each EM iteration. They

guarantee what we refer to as an $(\epsilon, \delta)$-*probably approximately monotonic* (PAM) update, stating that $Q(\widehat{\boldsymbol{\theta}}^{k+1}; \widehat{\boldsymbol{\theta}}^k) \geq Q(\widehat{\boldsymbol{\theta}}^k; \widehat{\boldsymbol{\theta}}^k) - \epsilon$, with probability at least $1 - \delta$.

Rather than resorting to asymptotic approximations, we take advantage of the specific form of $Q$ in our problem to obtain the finite-sample PAM result below. Because $\widehat{Q}(\widehat{\boldsymbol{\theta}}^{k+1}; \widehat{\boldsymbol{\theta}}^k)$ involves terms $\log \widehat{A}_{i,j}^k$ and $\log \widehat{\pi}_i^k$, in practice we bound $\widehat{A}_{i,j}^k$ and $\widehat{\pi}_i^k$ away from zero to ensure that $\widehat{Q}$ does not blow up. Specifically, we assume a small positive constant $\theta_{\min}$ so that $\widehat{A}_{i,j}^k \geq \theta_{\min}$ and $\widehat{\pi}_i^k \geq \theta_{\min}$.

**Theorem 1** *Let $\epsilon, \delta > 0$ be given. There exist finite constants $b_m > 0$, independent of $N_m$, so that if*

$$L_m = \frac{2 b_m^2 T^2 N_m^4 |\log \theta_{\min}|^2}{\epsilon^2} \log\left(\frac{2 N_m^2}{1 - (1 - \delta)^{1/T}}\right) \tag{9}$$

*importance samples are used for the $m$th observation, then $Q(\widehat{\boldsymbol{\theta}}^{k+1}; \widehat{\boldsymbol{\theta}}^k) \geq Q(\widehat{\boldsymbol{\theta}}^k; \widehat{\boldsymbol{\theta}}^k) - \epsilon$, with probability greater than $1 - \delta$.*

The proof involves two key steps. First, we derive finite sample concentration-style bounds for the importance sample estimates showing, *e.g.*, that $\widehat{\alpha}_{t',t''}^{(m)}$ converges to $\bar{\alpha}_{t',t''}^{(m)}$ at a rate which is exponential in the number of importance samples used. These bounds are based on rather novel concentration inequalities for importance sampling estimators, which may be of interest in their own right (see the supplementary document [7] for details). Then, accounting for the explicit form of $Q$ in our problem, the result follows from application of the union bound and the assumptions that $\widehat{A}_{i,j}^k, \widehat{\pi}_i^k \geq \theta_{\min}$. In fact, by making a slightly stronger assumption it can be shown that the MCEM update is *probably monotonic* (*i.e.*, $(0, \delta)$-PAM, not *approximately* monotonic) if $L_m'$ importance samples are used for the $m$th observation, where $L_m'$ also depends polynomially on $N_m$ and $T$. See the supplementary document [7] for further discussion and for the full proof of Theorem 1.

Recall that exact E-step computation requires $N_m!$ operations for the $m$th observation (enumerating all permutations). The bound above stipulates that the number of importance samples required for a PAM update is on the order of $N_m^4 \log N_m^2$. Generating one importance sample using the sequential procedure described above requires $N_m$ operations. In contrast to the (exponential complexity) exact EM algorithm, this clearly demonstrates that the MCEM converges with high probability while only having polynomial computational complexity, and, in this sense, the MCEM meaningfully breaks the curse of dimensionality by using randomness to preserve the monotonic convergence property.

## 4  Experimental Results

The performance of our algorithm for *network inference from co-occurrences* (NICO, pronounced "*nee-koh*") has been evaluated on both simulated data and on a biological data set. In these experiments, network structure is inferred by first executing the EM algorithm to infer the parameters $(\mathbf{A}, \boldsymbol{\pi})$ of a Markov chain. Then, inserting edges in the inferred graph based on the most likely order of each path according to $(\mathbf{A}, \boldsymbol{\pi})$ ensures the resulting graph is feasible with respect to the observations. Because the EM algorithm is only guaranteed to converge to a local maximum, we rerun the algorithm from multiple random initializations and chose the mostly likely of these solutions. To gauge the performance of our algorithm we use the *edge symmetric difference error*: the total number of false positives (edges in the inferred network which do not exist in the true network) plus the number of false negatives (edges in the true network not appearing in the inferred network).

We simulate co-occurrence observations in the following fashion. A random graph on 50 vertices is sampled. Disjoint sets of vertices are randomly chosen as path origins and destinations, paths are generated between each origin-destination pair using the shortest path algorithm with either unit weight per edge ("shortest path") or a random weight on each edge ("random routing"), and then co-occurrence observations are formed from each path. We keep the number of origins fixed at 5 and vary the number of destinations between 5 and 40 to see how the number of observations effects performance. NICO performance is compared against the *frequency method* (FM) described in [4].

Figure 1 plots the edge error for synthetic data generated using (a) shortest path routing, and (b) random routing. Each curve is the average performance over 100 different network and path real-

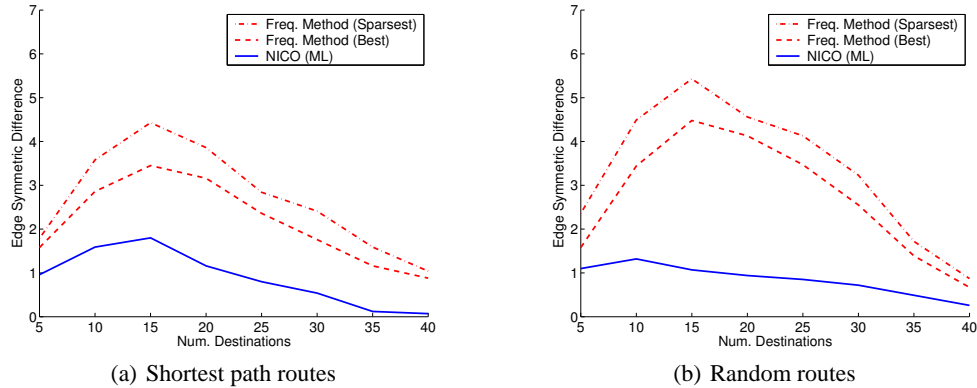

|   (a) Shortest path routes   |   (b) Random routes   |

Figure 1: Edge symmetric differences between inferred networks and the network one would obtain using co-occurrence measurements arranged in the correct order. Performance is averaged over 100 different network realizations. For each configuration 10 NICO and FM solutions are obtained via different initializations. We then choose the NICO solution yielding the largest likelihood, and compare with both the sparsest (fewest edges) and clairvoyant best (lowest error) FM solution.

izations. For each network/path realization, the EM algorithm is executed with 10 random initializations. Exact E-step calculation is used for observations with $N_m \leq 12$, and importance sampling is used for longer paths. The longest observation in our data has $N_m = 19$. The FM uses simple pairwise frequencies of co-occurrence to assign an order independently to each path observation. Of the 10 NICO solutions (different random initializations), we use the one based on parameter estimates yielding the highest likelihood score which also always gives the best performance. Because it is a heuristic, the FM does not provide a similar mechanism for ranking solutions from different initializations. We plot FM performance for two schemes; one based on choosing the sparsest FM solution (the one with the fewest edges), and one based on clairvoyantly choosing the FM solution with lowest error. NICO consistently outperforms even the clairvoyant best FM solution.

Our method has also been applied to infer the stress-activated protein kinase (SAPK)/Jun $N$-terminal kinase (JNK) and NF$\kappa$B signal transduction pathways[1] (biological networks). The clustering procedure described in [2] is applied to microarray data in order to identify 18 co-occurrences arising from different environmental stresses or growth factors (path source) and terminating in the production of SAPK/JNK or NF$\kappa$B proteins. The reconstructed network (combined SAPK/JNK and NF$\kappa$B signal transduction pathways) is depicted in Figure 2. This structure agrees with the signalling pathways identified using traditional experimental techniques which test individually for each possible edge (*e.g.*, "MAPK" and "NF-$\kappa$B Signaling" on http://www.cellsignal.com).

## 5 Conclusion

This paper describes a probabilistic model and statistical inference procedure for inferring network structure from incomplete "co-occurrence" measurements. Co-occurrences are modelled as samples of a first-order Markov chain subjected to a random permutation. We describe exact and Monte Carlo EM algorithms for calculating maximum likelihood estimates of the Markov chain parameters (initial state distribution and transition matrix), treating the random permutations as hidden variables. Standard results for the EM algorithm guarantee convergence to a local maximum. Although our exact EM algorithm has exponential computational complexity, we provide finite-sample bounds guaranteeing convergence of the Monte Carlo EM variation to a local maximum with high probability and with only polynomial complexity. Our algorithm is easily extended to compute maximum *a posteriori* estimates, applying a Dirichlet prior to the initial state distribution and to each row of the Markov transition matrix.

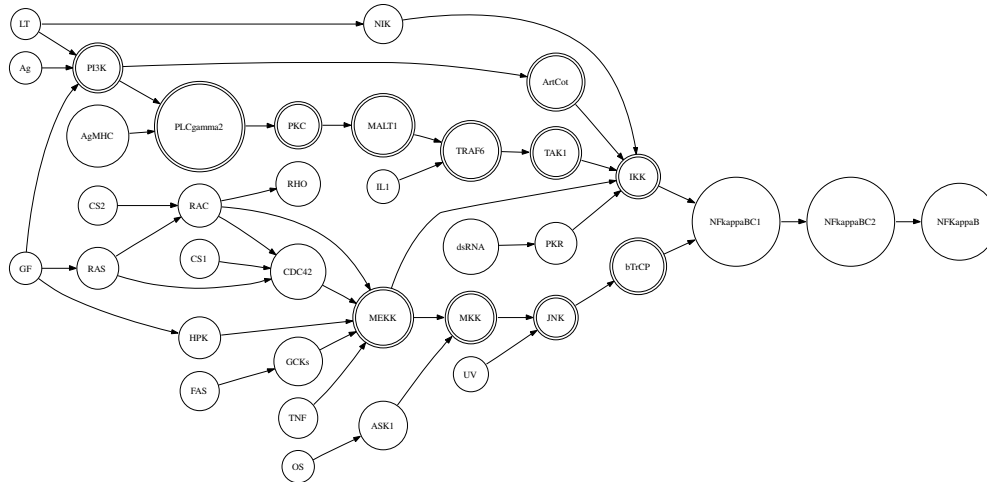

Figure 2: Inferred topology of the combined SAPK/JNK and NFκB signal transduction pathways. Co-occurrences are obtained from gene expression data via the clustering algorithm described in [2], and then network is inferred using NICO.

## Acknowledgments

The authors of this paper would like to thank D. Zhu and A.O. Hero for providing the data and collaborating on the biological network experiment reported in Section 4. This work was supported in part by the Portuguese Foundation for Science and Technology grant POSC/EEA-SRI/61924/2004, the Directorate of National Intelligence, and National Science Foundation grants CCF-0353079 and CCR-0350213.

## Footnotes

[1]NF$\kappa$B proteins control genes regulating a broad range of biological processes including innate and adaptive immunity, inflammation and B cell development. The NF$\kappa$B pathway is a collection of paths activated by various environmental stresses and growth factors, and terminating in the production of NF$\kappa$B.

## References

[1] E. Klipp, R. Herwig, A. Kowald, C. Wierling, and H. Lehrach. *Systems Biology in Practice: Concepts, Implementation and Application*. John Wiley & Sons, 2005.

[2] D. Zhu, A. O. Hero, H. Cheng, R. Khanna, and A. Swaroop. Network constrained clustering for gene microarray data. *Bioinformatics*, 21(21):4014–4020, 2005.

[3] Y. Liu and H. Zhao. A computational approach for ordering signal transduction pathway components from genomics and proteomics data. *BMC Bioinformatics*, 5(158), October 2004.

[4] M. G. Rabbat, J. R. Treichler, S. L. Wood, and M. G. Larimore. Understanding the topology of a telephone network via internally-sensed network tomography. In *Proc. IEEE International Conference on Acoustics, Speech, and Signal Processing*, 2005.

[5] O. Sporns and G. Tononi. Classes of network connectivity and dynamics. *Complexity*, 7(1):28–38, 2002.

[6] O. Sporns, D. R. Chialvo, M. Kaiser, and C. C. Hilgetag. Organization, development and function of complex brain networks. *Trends in Cognitive Science*, 8(9), 2004.

[7] M.G. Rabbat, M.A.T. Figueiredo, and R.D. Nowak. Supplement to inferring network structure from co-occurrences. Technical report, University of Wisconsin-Madison, October 2006.

[8] J. Kubica, A. Moore, D. Cohn, and J. Schneider. cGraph: A fast graph-based method for link analysis and queries. In *Proc. IJCAI Text-Mining and Link-Analysis Workshop*, Acapulco, Mexico, August 2003.

[9] D. Heckerman, D. Geiger, and D. Chickering. Learning Bayesian networks: The combination of knowledge and statistical data. *Machine Learning*, 20:197–243, 1995.

[10] N. Friedman and D. Koller. Being Bayesian about Bayesian network structure: A Bayesian approach to structure discovery in Bayesian networks. *Machine Learning*, 50(1–2):95–125, 2003.

[11] R. A. Boyles. On the convergence of the EM algorithm. *J. Royal Statistical Society B*, 45(1):47–50, 1983.

[12] C. F. J. Wu. On the convergence properties of the EM algorithm. *Ann. of Statistics*, 11(1):95–103, 1983.

[13] C. Robert and G. Casella. *Monte Carlo Statistical Methods*. Springer Verlag, New York, 1999.

[14] B. S. Caffo, W. Jank, and G. L. Jones. Ascent-based Monte Carlo EM. *J. Royal Statistical Society B*, 67(2):235–252, 2005.
